# Direction Selectivity In Primary Visual Cortex Using Massive Intracortical Connections

**Humbert Suarez**
CNS Program 216-76
Caltech
Pasadena, CA 91125

**Christof Koch**
CNS Program 216-76
Caltech
Pasadena, CA 91125

**Rodney Douglas**
MRC Anatomical Neuropharmacology Unit
University of Oxford
Oxford
UK

## Abstract

Almost all models of orientation and direction selectivity in visual cortex are based on feedforward connection schemes, where geniculate input provides all excitation to both pyramidal and inhibitory neurons. The latter neurons then suppress the response of the former for non-optimal stimuli. However, anatomical studies show that up to 90 % of the excitatory synaptic input onto any cortical cell is provided by other cortical cells. The massive excitatory feedback nature of cortical circuits is embedded in the *canonical microcircuit* of Douglas & Martin (1991). We here investigate analytically and through biologically realistic simulations the functioning of a detailed model of this circuitry, operating in a *hysteretic* mode. In the model, weak geniculate input is dramatically amplified by intracortical excitation, while inhibition has a dual role: (i) to prevent the early geniculate-induced excitation in the null direction and (ii) to restrain excitation and ensure that the neurons fire only when the stimulus is in their receptive-field. Among the

insights gained are the possibility that hysteresis underlies visual cortical function, paralleling proposals for short-term memory, and strong limitations on linearity tests that use gratings. Properties of visual cortical neurons are compared in detail to this model and to a classical model of direction selectivity that does not include excitatory cortico-cortical connections. The model explain a number of puzzling features of direction-selective simple cells, including the small somatic input conductance changes that have been measured experimentally during stimulation in the null direction. The model also allows us to understand why the velocity-response curve of area 17 neurons is different from that of their LGN afferents, and the origin of expansive and compressive nonlinearities in the contrast-response curve of striate cortical neurons.

# 1  INTRODUCTION

Direction selectivity is the property of neurons that fire more strongly for one direction of motion of a bar (the preferred direction) than the other (null direction). It is one of the fundamental properties of neurons in visual cortex and is intimately related to the processing of motion by the visual system. LGN neurons that provide input to visual cortex respond approximately symmetrically to motion in different directions; so cortical neurons must generate that direction specificity. Models of direction selectivity in primary visual cortex generally overlook two important constraints. **1.** at least 80% of excitatory synapses on cortical pyramidal cells originate from other pyramidal cells, and less than 10 % are thalamic afferents (Peters & Payne, 1993). **2.** Intracellular *in vivo* recordings in cat simple cells by Douglas, et al. (1988) failed to detect significant changes in somatic input conductance during stimulation in the null direction, indicating that there is very little synaptic input to direction-selective neurons in that condition, including no massive "shunting" inhibition.

One very attractive solution incorporating these two constraints was proposed by Douglas & Martin (1991) in the form of their *canonical microcircuit*: for motion of a visual stimulus in the preferred direction, weak geniculate excitation excites cortical pyramidal cells to respond moderately. This relatively small amount of cortical excitation is amplified via excitatory cortico-cortical connections. For motion in the null direction, the weak geniculate excitation is vetoed by weak inhibition (mediated via an interneuron) and the cortical loop is never activated. In order to test quantitatively this circuit against the large body of experimental data, it is imperative to model its operation through mathematical analysis and detailed simulations.

# 2  MODEL DESCRIPTION AND ANALYSIS

The model consists of a retino-geniculate and a cortical stage (Fig. 1). The former includes a center-surround receptive field and bandpass temporal filtering (Victor, 1987). We simulate a 1-D array of ON LGN neurons, with 208 LGN neurons at each of 6 spatial positions in this array. The output of the LGN—action potentials—feeds

into the cortical module consisting of 640 pyramidal (excitatory) and 160 inhibitory neurons. Each neuron is modeled using 3-4 compartments whose parameters reflect, in a simplified way, the biophysics and morphology of cortical neurons; the somatic compartment produces action potentials in response to current injection. There are excitatory connections among all pyramidal neurons, and inhibitory connections from the inhibitory population to itself and to the pyramidal population. The receptive field of the geniculate input to the inhibitory cortical neurons is displaced in space from the geniculate input to the pyramidal neurons, so that in the null direction inhibition overlaps with geniculate excitation in the pyramidal neurons, resulting in direction selectivity in the pyramidal neurons. The time courses of the post-synaptic potentials (PSP's) are consistent with physiologically recorded PSP's. The EPSP's in our model arise exclusively from non-NMDA synapses, and the IPSP's originate from both $GABA_A$ and $GABA_B$ synapses.

There are two operating modes of this cortical amplifier circuit, depending on parameter values. In the first mode, the pyramidal neurons' response increases proportionally to the stimulus strength over a substantial range of input values, before saturating. In the second mode, the response increases much faster over a narrow range of stimulus strengths, then saturates. Analytically, one can define a steady-state transfer function for the network of pyramidal neurons; then, in the first mode, the transfer function has a slope that is less than 1, so that the network's firing rate at equilibrium increases proportionally to the input strength and the network dynamics are rather slow. In the second mode the initial slope is larger than 1, so that the network can discharge briskly at equilibrium even without any input, show hysteresis, and has rather fast dynamics. The network does not fire without any input, because of the neuron's threshold. In this paper, we will show responses in that second, hysteretic mode of operation.

We will compare the cortical amplifier model's response properties to a pure feed-forward model, that has no excitatory connections between pyramidal neurons. In order to maintain strong responses in that model, the LGN was connected more strongly to the pyramidal neurons, and in order to maintain direction selectivity, the weights of the connections of the inhibitory neurons to the pyramids were increased as well.

## 3 AMPLIFICATION AND CONDUCTANCE CHANGE IN THE NULL DIRECTION

The input conductance of pyramidal neurons, a measure of total synaptic input, changes by only 50 % in the cortical amplifier model versus 400 % in the feedforward model, and so is more consistent with physiology (Fig. 2). Indeed, most of the current causing firing in the preferred direction originates from other pyramidal neurons, so the connection weight from the LGN is small. Consequently, the inhibitory weight is also small, since it needs only be large enough to balance out the LGN current in the null direction. Since there is little firing in other pyramidal neurons in the null direction, there is also little total synaptic input to the fiducial cell. The cortical amplifier circuit provides substantial amplification of the LGN input. In the preferred direction, excitatory intracortical connections amplify the LGN input, providing a feedback current that is about 2.2 times larger than the

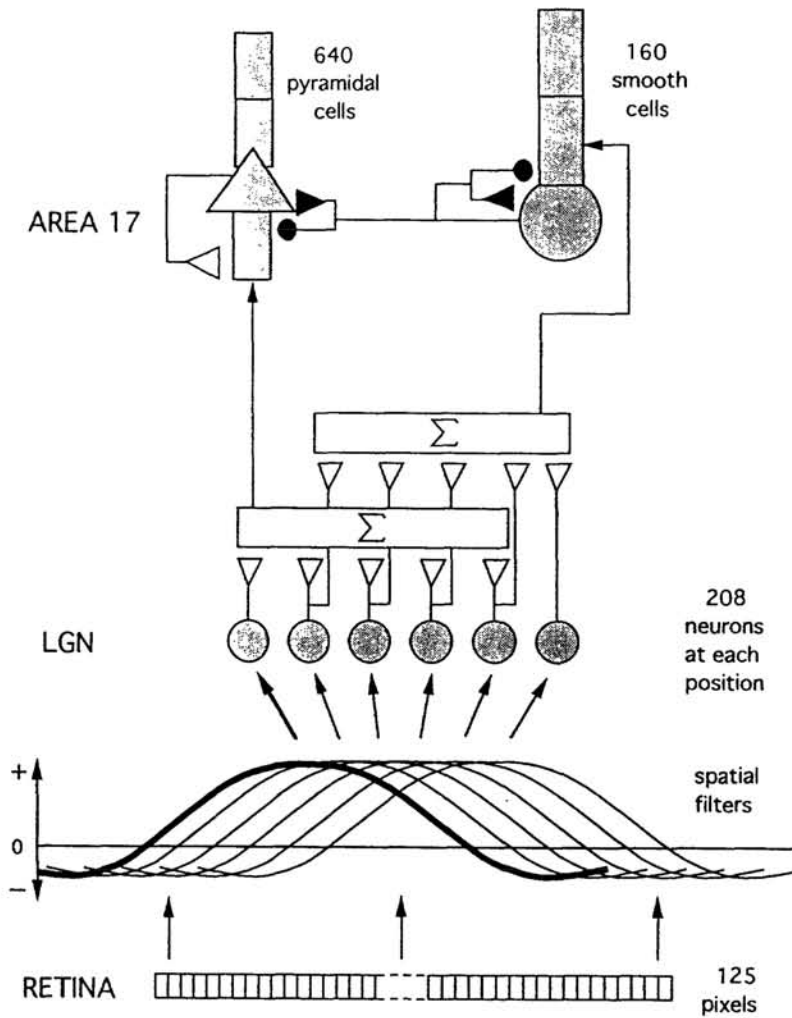

Figure 1: Wiring diagram of the direction selectivity model. Input to LGN neurons comes from a one-dimensional array of retinal pixels. There are 208 LGN neurons at each of six spatial positions. The LGN neurons connect slightly differently with the two populations of cortical neurons (pyramidal and inhibitory) so that as a group the LGN inputs to the pyramids are displaced spatially by 5′ from those to the inhibitory neurons. The open triangle symbols denote excitatory connections, the filled triangles inhibitory $GABA_A$-mediated synapses, and the filled circles inhibitory $GABA_B$-mediated synapses. The capital sigma symbol indicates convergence of inputs from many LGN neurons onto cortical neurons.

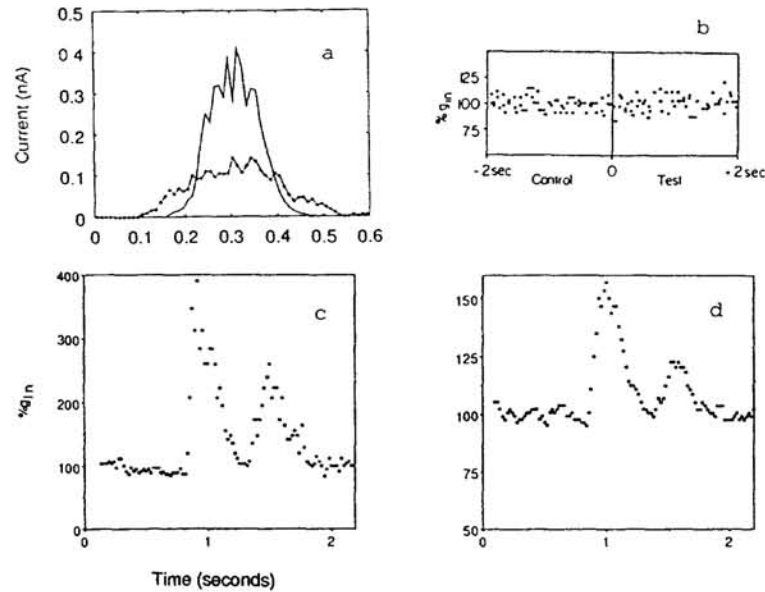

Figure 2: (a) Total synaptic current in a simulated pyramidal cell from the LGN (with symbols) and from other pyramidal cells (continuous without symbols), during stimulation by a bar moving in the preferred direction. For the same stimulus moving in the null direction, somatic input conductance as a function of time, (b) for a direction-selective pyramidal neuron (data from Douglas *et al.*, 1988); (c) feedforward model; (d) cortical amplifier model;

LGN current at 60 % contrast (Fig. 2).

## 4   CONTRAST-RESPONSE CURVES

Contrast-response curves plot the peak firing rate to a grating moving in the preferred direction as function of its contrast, or stimulus amplitude (Albrecht & Hamilton, 1982). The cortical amplifier's contrast-response curve is very different from the LGN inputs' and is similar to those that have been described experimentally in cortex (Albrecht & Geisler 1991), having a steep power-function portion followed by abrupt saturation (Fig. 3). The network firing saturates at the fixed point of the transfer function mentioned above (see section 2) and the steep portion results from the fast rise to that fixed point when the stimulus has exceeded the neurons' threshold. In contrast, the feedforward model's contrast-response curve is similar to the LGN inputs' and does not match physiology. The response is very small in the null direction, resulting in very good direction selectivity at all contrasts (i.e., the average direction index is above 0.9).

## 5   VELOCITY-RESPONSE CURVES

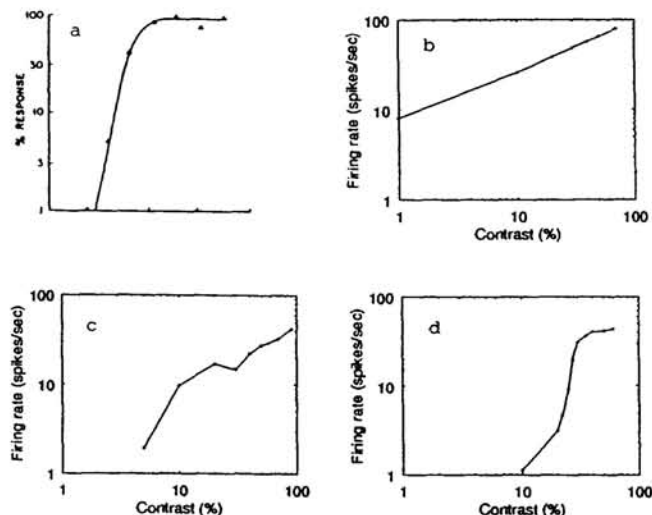

Figure 3: Peak firing rate versus contrast during stimulation by a moving grating.
(a) Visual cortical neuron. Data from Albrecht & Hamilton, 1982. (b) LGN model.
(c) Feedforward model. (d) Cortical amplifier model.

Velocity-response curves plot the peak response to a bar moving in the preferred
direction as a function of its velocity . Again, the cortical amplifier's velocity-
response curve is very different from the LGN inputs' and is similar to physiology
(Orban, 1984; Fig. 4). The LGN model is firing strongly at high velocities but
the model pyramidal neurons are totally silent, due to a combination of $GABA_A$
inhibition, neuron threshold, and membrane low-pass filtering. At low velocities,
the LGN model does not fire much, while the cortical neurons respond strongly.
Indeed, the network firing has enough time to reach the fixed point of the transfer
function and will reach it as long as the input is suprathreshold. In contrast, the
feedforward model's velocity-response curve is again similar to the LGN input's
and does not match physiology. Also shown in Fig. 4 is the response in the null
direction for the cortical amplifier model. There is very good direction selectivity at
all velocities, consistent with physiological data (Orban, 1984). The persistence of
direction selectivity down to low velocities depends critically on the time constant
of $GABA_B$ and the presence of a very small displacement between the LGN inputs
to the pyramidal and inhibitory neurons.

## 6   OTHER PROPERTIES

Recently, direction-selective cortical neurons have been tested for linearity using
an intracellular grating superposition test and found to be quite linear (Jagadeesh
et al., 1993). Despite that amplification in the present model is so nonlinear, the
model is also linear according to that superposition test. An analysis of the test in
the context of the model shows that such a test has limited usefulness and suggests
improvements.

Given that the network transfer function has a fixed point at high firing without any

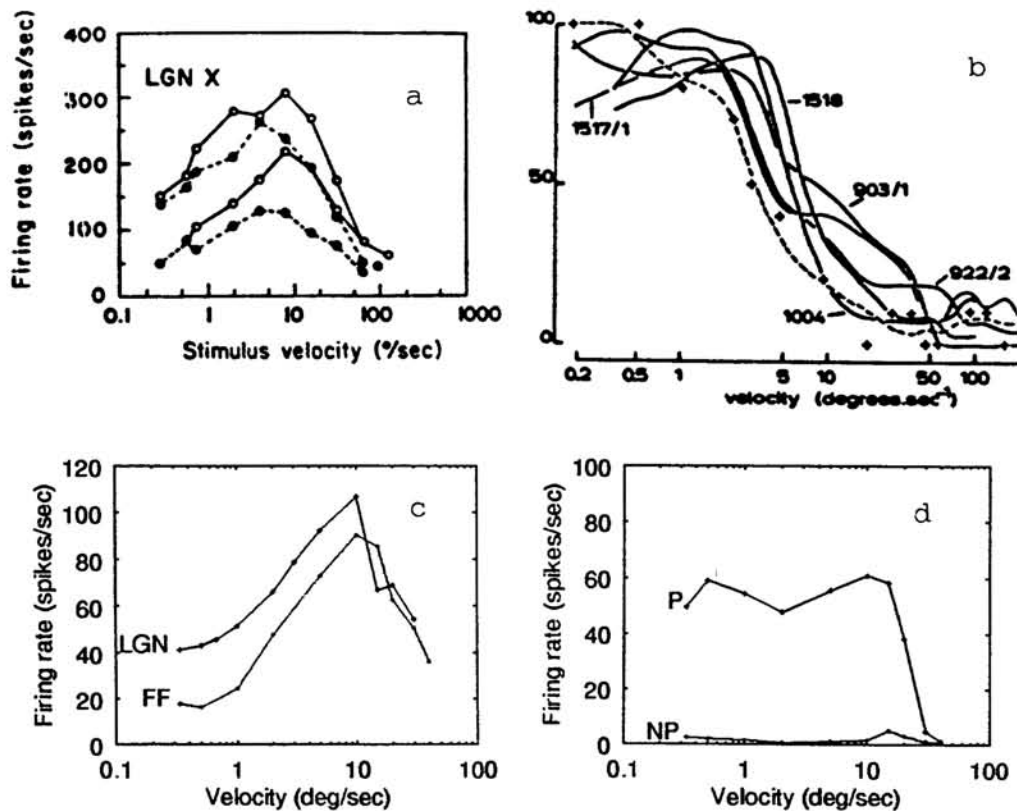

Figure 4: Peak firing rate versus velocity during stimulation by a bar. (a) LGN neuron in cat (Frishman *et al.*, 1983). (b) area 17 visual cortical neuron in cat (Orban, 1984). (c) LGN model neuron (curve labelled "LGN") and feedforward model (FF). (d) Cortical amplifier model in the preferred (curve labelled "P") and null directions (NP).

input (see section 2), *hysteresis* may occur, whereby the network's discharge persists after the initial stimulus is withdrawn. Because of hysteresis, it is imperative to *reset* the network by presenting a negative, or inhibitory, input. A parallel can be drawn to several recent proposals for the mechanisms underlying short-term memory.

# 7  CONCLUSIONS

From early on, neurophysiologists have proposed that the LGN provides most of the input to visual cortical neurons and shapes their receptive properties (Hubel and Wiesel, 1962). However, direction selectivity and several other stimulus selectivities are not present in LGN neurons, and other important discrepancies have appeared between the receptive field properties of cortical neurons and those of their LGN afferents. Anatomically, synaptic inputs from the LGN account for less than 10 % of synapses to pyramidal neurons in visual cortex; the remaining 90 % could clearly provide a substrate for these receptive field transformations. Although intracortical inhibition has often been invoked to explain various cortical properties, excitation is usually not mentioned, despite that most cortico-cortical synapses are excitatory. In this paper, we show that intracortical excitation can better account for several key properties of cortical neurons than a purely feedforward model, including the magnitude of the conductance change in the null direction, contrast-response curves, and velocity-response curves. Furthermore, surprisingly, other key cell properties that are appear to point to feedforward models, such as linearity measured by superposition tests, are also properties of a model based on intracortical excitation.

### Acknowledgements

This research was supported by the Office of Naval Research, the National Science Foundation, the National Eye Institute, and the McDonnell Foundation.

### References

Albrecht, D.G., and Geisler, W.S. (1991) *Visual Neuroscience* **7**, 531-546.
Albrecht, D.B., and Hamilton, D.B. (1982) *J. Neurophysiol.* **48**, 217-237.
Douglas, R.J., and Martin, K.A.C. (1991) *J. Physiol.* **440**, 735-769.
Douglas, R.J., Martin, K.A.C., and Whitteridge, D. (1988) *Nature* **332**, 642-644.
Frishman, L.J., Schweitzer-Tong, D.E., and Goldstein, E.B. (1983) *J. Neurophysiol.* **50**, 1393-1414.
Hubel, D.H., and Wiesel, T.N. (1962) *J. Physiol.* **165**, 559-568.
Jagadeesh, B., Wheat, H.S. and Ferster, D. (1993) *Science* **262**, 1901-1904
Orban, G.A. (1984) Neuronal operations in the visual cortex. Springer, Berlin.
Peters, A. and Payne, B. R. (1993) *Cerebral Cortex* **3**, 69-78.
Victor, J.D. (1987) *J.Physiol.* **386**, 219-246.